# Bayesian Model Comparison
# by Monte Carlo Chaining

**David Barber**
D.Barber@aston.ac.uk

**Christopher M. Bishop**
C.M.Bishop@aston.ac.uk

Neural Computing Research Group
Aston University, Birmingham, B4 7ET, U.K.
http://www.ncrg.aston.ac.uk/

## Abstract

The techniques of Bayesian inference have been applied with great success to many problems in neural computing including evaluation of regression functions, determination of error bars on predictions, and the treatment of hyper-parameters. However, the problem of model comparison is a much more challenging one for which current techniques have significant limitations. In this paper we show how an extended form of Markov chain Monte Carlo, called *chaining*, is able to provide effective estimates of the relative probabilities of different models. We present results from the robot arm problem and compare them with the corresponding results obtained using the standard Gaussian approximation framework.

## 1 Bayesian Model Comparison

In a Bayesian treatment of statistical inference, our state of knowledge of the values of the parameters $\mathbf{w}$ in a model $\mathcal{M}$ is described in terms of a probability distribution function. Initially this is chosen to be some prior distribution $p(\mathbf{w}|\mathcal{M})$, which can be combined with a likelihood function $p(D|\mathbf{w}, \mathcal{M})$ using Bayes' theorem to give a posterior distribution $p(\mathbf{w}|D, \mathcal{M})$ in the form

$$p(\mathbf{w}|D, \mathcal{M}) = \frac{p(D|\mathbf{w}, \mathcal{M})p(\mathbf{w}|\mathcal{M})}{p(D|\mathcal{M})} \tag{1}$$

where $D$ is the data set. Predictions of the model are obtained by performing integrations weighted by the posterior distribution.

The comparison of different models $\mathcal{M}_i$ is based on their relative probabilities, which can be expressed, again using Bayes' theorem, in terms of prior probabilities $p(\mathcal{M}_i)$ to give

$$\frac{p(\mathcal{M}_i|D)}{p(\mathcal{M}_j|D)} = \frac{p(D|\mathcal{M}_i)p(\mathcal{M}_i)}{p(D|\mathcal{M}_j)p(\mathcal{M}_j)} \tag{2}$$

and so requires that we be able to evaluate the *model evidence* $p(D|\mathcal{M}_i)$, which corresponds to the denominator in (1). The relative probabilities of different models can be used to select the single most probable model, or to form a committee of models, weighed by their probabilities.

It is convenient to write the numerator of (1) in the form $\exp\{-E(\mathbf{w})\}$, where $E(\mathbf{w})$ is an error function. Normalization of the posterior distribution then requires that

$$p(D|\mathcal{M}) = \int \exp\{-E(\mathbf{w})\}\,d\mathbf{w}. \tag{3}$$

Generally, it is straightforward to evaluate $E(\mathbf{w})$ for a given value of $\mathbf{w}$, although it is extremely difficult to evaluate the corresponding model evidence using (3) since the posterior distribution is typically very small except in narrow regions of the high-dimensional parameter space, which are unknown a-priori. Standard numerical integration techniques are therefore inapplicable.

One approach is based on a local Gaussian approximation around a mode of the posterior (MacKay, 1992). Unfortunately, this approximation is expected to be accurate only when the number of data points is large in relation to the number of parameters in the model. In fact it is for relatively complex models, or problems for which data is scarce, that Bayesian methods have the most to offer. Indeed, Neal (1996) has argued that, from a Bayesian perspective, there is no reason to limit the number of parameters in a model, other than for computational reasons. We therefore consider an approach to the evaluation of model evidence which overcomes the limitations of the Gaussian framework. For additional techniques and references to Bayesian model comparison, see Gilks *et al.* (1995) and Kass and Raftery (1995).

## 2  Chaining

Suppose we have a simple model $\mathcal{M}_0$ for which we can evaluate the evidence analytically, and for which we can easily generate a sample $\mathbf{w}^l$ (where $l = 1, \ldots, L$) from the corresponding distribution $p(\mathbf{w}|D, \mathcal{M}_0)$. Then the evidence for some other model $\mathcal{M}$ can be expressed in the form

$$\begin{aligned}
\frac{p(D|\mathcal{M})}{p(D|\mathcal{M}_0)} &= \int \exp\{-E(\mathbf{w}) + E_0(\mathbf{w})\}p(\mathbf{w}|D, \mathcal{M}_0)\,d\mathbf{w} \\
&\simeq \frac{1}{L}\sum_{l=1}^{L}\exp\{-E(\mathbf{w}^l) + E_0(\mathbf{w}^l)\}.
\end{aligned} \tag{4}$$

Unfortunately, the Monte Carlo approximation in (4) will be poor if the two error functions are significantly different, since the exponent is dominated by regions where $E$ is relatively small, for which there will be few samples unless $E_0$ is also small in those regions. A simple Monte Carlo approach will therefore yield poor results. This problem is equivalent to the evaluation of free energies in statistical physics,

which is known to be a challenging problem, and where a number of approaches have been developed Neal (1993).

Here we discuss one such approach to this problem based on a chain of $K$ successive models $\mathcal{M}_i$ which interpolate between $\mathcal{M}_0$ and $\mathcal{M}$, so that the required evidence can be written as

$$p(D|\mathcal{M}) = p(D|\mathcal{M}_0) \frac{p(D|\mathcal{M}_1)}{p(D|\mathcal{M}_0)} \frac{p(D|\mathcal{M}_2)}{p(D|\mathcal{M}_1)} \cdots \frac{p(D|\mathcal{M})}{p(D|\mathcal{M}_K)}. \qquad (5)$$

Each of the ratios in (5) can be evaluated using (4). The goal is to devise a chain of models such that each successive pair of models has probability distributions which are reasonably close, so that each of the ratios in (5) can be evaluated accurately, while keeping the total number of links in the chain fairly small to limit the computational costs.

We have chosen the technique of hybrid Monte Carlo (Duane *et al.*, 1987; Neal, 1993) to sample from the various distributions, since this has been shown to be effective for sampling from the complex distributions arising with neural network models (Neal, 1996). This involves introducing Hamiltonian equations of motion in which the parameters $\mathbf{w}$ are augmented by a set of fictitious 'momentum' variables, which are then integrated using the leapfrog method. At the end of each trajectory the new parameter vector is accepted with a probability governed by the Metropolis criterion, and the momenta are replaced using Gibbs sampling. As a check on our software implementation of chaining, we have evaluated the evidence for a mixture of two non-isotropic Gaussian distributions, and obtained a result which was within 10% of the analytical solution.

## 3 Application to Neural Networks

We now consider the application of the chaining method to regression problems involving neural network models. The network corresponds to a function $y(\mathbf{x}, \mathbf{w})$, and the data set consists of $N$ pairs of input vectors $\mathbf{x}_n$ and corresponding targets $\mathbf{t}_n$ where $n = 1, \ldots, N$. Assuming Gaussian noise on the target data, the likelihood function takes the form

$$p(D|\mathbf{w}, \mathcal{M}) = \left(\frac{\beta}{2\pi}\right)^{N/2} \exp\left\{-\frac{\beta}{2}\sum_{n=1}^{N} \|y(\mathbf{x}_n; \mathbf{w}) - \mathbf{t}_n\|^2\right\} \qquad (6)$$

where $\beta$ is a hyper-parameter representing the inverse of the noise variance. We consider networks with a single hidden layer of 'tanh' units, and linear output units. Following Neal (1996) we use a diagonal Gaussian prior in which the weights are divided into groups $\mathbf{w}_k$, where $k = 1, \ldots, 4$ corresponding to input-to-hidden weights, hidden-unit biases, hidden-to-output weights, and output biases. Each group is governed by a separate 'precision' hyper-parameter $\alpha_k$, so that the prior takes the form

$$p(\mathbf{w}|\{\alpha_k\}) = \frac{1}{Z_W} \exp\left\{-\frac{1}{2}\sum_k \alpha_k \mathbf{w}_k^{\mathrm{T}} \mathbf{w}_k\right\} \qquad (7)$$

where $Z_W$ is the normalization coefficient. The hyper-parameters $\{\alpha_k\}$ and $\beta$ are themselves each governed by hyper-priors given by Gamma distributions of the form

$$p(\alpha) \propto \alpha^s \exp(-\alpha s/2\omega) \qquad (8)$$

in which the mean $\omega$ and variance $2\omega^2/s$ are chosen to give very broad hyper-priors in reflection of our limited prior knowledge of the values of the hyper-parameters. We use the hybrid Monte Carlo algorithm to sample from the joint distribution of parameters and hyper-parameters. For the evaluation of evidence ratios, however, we consider only the parameter samples, and perform the integrals over hyper-parameters analytically, using the fact that the gamma distribution is conjugate to the Gaussian.

In order to apply chaining to this problem, we choose the prior as our reference distribution, and then define a set of intermediate distributions based on a parameter $\lambda$ which governs the effective contribution from the data term, so that

$$E(\lambda, \mathbf{w}) = \lambda\phi(\mathbf{w}) + E_0(\mathbf{w}) \tag{9}$$

where $\phi(\mathbf{w})$ arises from the likelihood term (6) while $E_0(\mathbf{w})$ corresponds to the prior (7). We select a set of 18 values of $\lambda$ which interpolate between the reference distribution ($\lambda = 0$) and the desired model distribution ($\lambda = 1$). The evidence for the prior alone is easily evaluated analytically.

## 4   Gaussian Approximation

As a comparison against the method of chaining, we consider the framework of MacKay (1992) based on a local Gaussian approximation to the posterior distribution. This approach makes use of the *evidence approximation* in which the integration over hyper-parameters is approximated by setting them to specific values which are themselves determined by maximizing their evidence functions.

This leads to a hierarchical treatment as follows. At the lowest level, the maximum $\widehat{\mathbf{w}}$ of the posterior distribution over weights is found for fixed values of the hyper-parameters by minimizing the error function. Periodically the hyper-parameters are re-estimated by evidence maximization, where the evidence is obtained analytically using the Gaussian approximation. This gives the following re-estimation formulae

$$\frac{1}{\beta} := \frac{1}{N-\gamma}\sum_{n=1}^{N}\|y(\mathbf{x}_n;\widehat{\mathbf{w}}) - t_n\|^2 \qquad \alpha_k := \frac{\gamma_k}{\widehat{\mathbf{w}}_k^T\widehat{\mathbf{w}}_k} \tag{10}$$

where $\gamma_k = W_k - \alpha_k\mathrm{Tr}_k(\mathbf{A}^{-1})$, $W_k$ is the total number of parameters in group $k$, $\mathbf{A} = \nabla\nabla E(\widehat{\mathbf{w}})$, $\gamma = \sum_k \gamma_k$, and $\mathrm{Tr}_k(\cdot)$ denotes the trace over the $k$th group of parameters. The weights are updated in an inner loop by minimizing the error function using a conjugate gradient optimizer, while the hyper-parameters are periodically re-estimated using (10)[1].

Once training is complete, the model evidence is evaluated by making a Gaussian approximation around the converged values of the hyper-parameters, and integrating over this distribution analytically. This gives the model log evidence as

$$\ln p(D|\mathcal{M}) = -E(\widehat{\mathbf{w}}) - \frac{1}{2}\ln|\mathbf{A}| + \frac{1}{2}\sum_k W_k \ln\alpha_k +$$
$$\frac{N}{2}\ln\beta + \ln h! + 2\ln h + \frac{1}{2}\sum_k \ln(2/\gamma_k) + \frac{1}{2}\ln(2/(N-\gamma)). \tag{11}$$

Here $h$ is the number of hidden units, and the terms $\ln h! + 2\ln h$ take account of the many equivalent modes of the posterior distribution arising from sign-flip and hidden unit interchange symmetries in the network model. A derivation of these results can be found in Bishop (1995; pages 434–436).

The result (11) corresponds to a single mode of the distribution. If we initialize the weight optimization algorithm with different random values we can find distinct solutions. In order to compute an overall evidence for the particular network model with a given number of hidden units, we make the assumption that we have found all of the distinct modes of the posterior distribution precisely once each, and then sum the evidences to arrive at the total model evidence. This neglects the possibility that some of the solutions found are related by symmetry transformations (and therefore already taken into account) or that we have missed important modes. While some attempt could be made to detect degenerate solutions, it will be difficult to do much better than the above within the framework of the Gaussian approximation.

## 5 Results: Robot Arm Problem

As an illustration of the evaluation of model evidence for a larger-scale problem we consider the modelling of the forward kinematics for a two-link robot arm in a two-dimensional space, as introduced by MacKay (1992). This problem was chosen as MacKay reports good results in using the Gaussian approximation framework to evaluate the evidences, and provides a good opportunity for comparison with the chaining approach. The task is to learn the mapping $(x_1, x_2) \rightarrow (y_1, y_2)$ given by

$$y_1 = 2.0\cos(x_1) + 1.3\cos(x_1 + x_2) \qquad y_2 = 2.0\sin(x_1) + 1.3\sin(x_1 + x_2) \quad (12)$$

where the data set consists of 200 input-output pairs with outputs corrupted by zero mean Gaussian noise with standard deviation $\sigma = 0.05$. We have used the original training data of MacKay, but generated our own test set of 1000 points using the same prescription. The evidence is evaluated using both chaining and the Gaussian approximation, for networks with various numbers of hidden units.

In the chaining method, the particular form of the gamma priors for the precision variables are as follows: for the input-to-hidden weights and hidden-unit biases, $\omega = 1$, $s = 0.2$; for the hidden-to-output weights, $\omega = h$, $s = 0.2$; for the output biases, $\omega = 0.2$, $s = 1$. The noise level hyper-parameters were $\omega = 400$, $s = 0.2$. These settings follow closely those used by Neal (1996) for the same problem. The hidden-to-output precision scaling was chosen by Neal such that the limit of an infinite number of hidden units is well defined and corresponds to a Gaussian process prior. For each evidence ratio in the chain, the first 100 samples from the hybrid Monte Carlo run, obtained with a trajectory length of 50 leapfrog iterations, are omitted to give the algorithm a chance to reach the equilibrium distribution. The next 600 samples are obtained using a trajectory length of 300 and are used to evaluate the evidence ratio.

In Figure 1 (a) we show the error values of the sampling stage for 24 hidden units, where we see that the errors are largely uncorrelated, as required for effective Monte Carlo sampling. In Figure 1 (b), we plot the values of $\ln\{p(D|\mathcal{M}_i)/p(D|\mathcal{M}_{i-1})\}$ against $\lambda_i$ $i = 1..18$. Note that there is a large change in the evidence ratios at the beginning of the chain, where we sample close to the reference distribution. For this

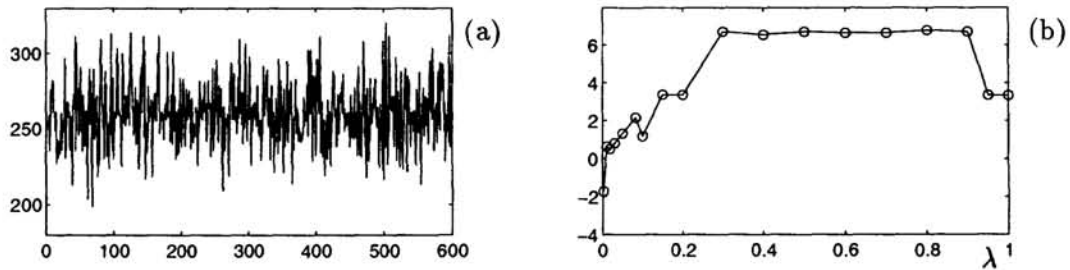

Figure 1: (a) error $E(\lambda = 0.6, \mathbf{w})$ for $h = 24$, plotted for 600 successive Monte Carlo samples. (b) Values of the ratio $\ln\{p(D|\mathcal{M}_i)/p(D|\mathcal{M}_{i-1})\}$ for $i = 1, \ldots, 18$ for $h = 24$.

reason, we choose the $\lambda_i$ to be dense close to $\lambda = 0$. We are currently researching more principled approaches to the partitioning selection. Figure 2 (a) shows the log model evidence against the number of hidden units. Note that the chaining approach is computationally expensive: for $h=24$, a complete chain takes 48 hours in a Matlab implementation running on a Silicon Graphics Challenge L.

We see that there is no decline in the evidence as the number of hidden units grows. Correspondingly, in Figure 2 (b), we see that the test error performance does not degrade as the number of hidden units increases. This indicates that there is no over-fitting with increasing model complexity, in accordance with Bayesian expectations.

The corresponding results from the Gaussian approximation approach are shown in Figure 3. We see that there is a characteristic 'Occam hill' whereby the evidence shows a peak at around $h = 12$, with a strong decrease for smaller values of $h$ and a slower decrease for larger values. The corresponding test set errors similarly show a minimum at around $h = 12$, indicating that the Gaussian approximation is becoming increasingly inaccurate for more complex models.

## 6  Discussion

We have seen that the use of chaining allows the effective evaluation of model evidences for neural networks using Monte Carlo techniques. In particular, we find that there is no peak in the model evidence, or the corresponding test set error, as the number of hidden units is increased, and so there is no indication of over-fitting. This is in accord with the expectation that model complexity should not be limited by the size of the data set, and is in marked contrast to the conventional

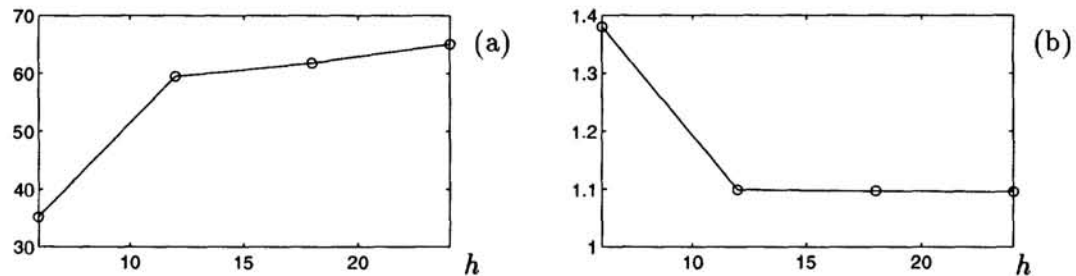

Figure 2: (a) Plot of $\ln p(D|\mathcal{M})$ for different numbers of hidden units. (b) Test error against the number of hidden units. Here the theoretical minimum value is 1.0. For $h = 64$ the test error is 1.11

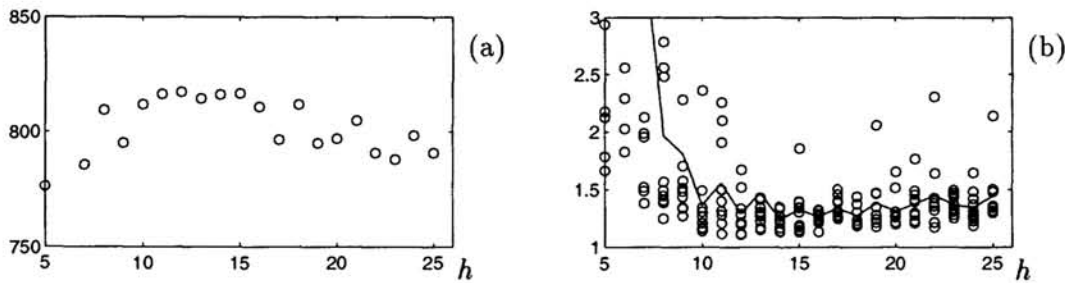

Figure 3: (a) Plot of the model evidence for the robot arm problem versus the number of hidden units, using the Gaussian approximation framework. This clearly shows the characteristic 'Occam hill' shape. Note that the evidence is computed up to an additive constant, and so the origin of the vertical axis has no significance. (b) Corresponding plot of the test set error versus the number of hidden units. Individual points correspond to particular modes of the posterior weight distribution, while the line shows the mean test set error for each value of $h$.

maximum likelihood viewpoint. It is also consistent with the result that, in the limit of an infinite number of hidden units, the prior over network weights leads to a well-defined Gaussian prior over functions (Williams, 1997).

An important advantage of being able to make accurate evaluations of the model evidence is the ability to compare quite distinct kinds of model, for example radial basis function networks and multi-layer perceptrons. This can be done either by chaining both models back to a common reference model, or by evaluating normalized model evidences explicitly.

## Acknowledgements

We would like to thank Chris Williams and Alastair Bruce for a number of useful discussions. This work was supported by EPSRC grant GR/J75425: *Novel Developments in Learning Theory for Neural Networks*.

## Footnotes

[1]Note that we are assuming that the hyper-priors (8) are sufficiently broad that they have no effect on the location of the evidence maximum and can therefore be neglected.

## References

Bishop, C. M. (1995). *Neural Networks for Pattern Recognition.* Oxford University Press.

Duane, S., A. D. Kennedy, B. J. Pendleton, and D. Roweth (1987). Hybrid Monte Carlo. *Physics Letters B* **195** (2), 216–222.

Gilks, W. R., S. Richardson, and D. J. Spiegelhalter (1995). *Markov Chain Monte Carlo in Practice.* Chapman and Hall.

Kass, R. E. and A. E. Raftery (1995). Bayes factors. *J. Am. Statist. Ass.* **90**, 773–795.

MacKay, D. J. C. (1992). A practical Bayesian framework for back-propagation networks. *Neural Computation* **4** (3), 448–472.

Neal, R. M. (1993). Probabilistic inference using Markov chain Monte Carlo methods. Technical Report CRG-TR-93-1, Department of Computer Science, University of Toronto, Cananda.

Neal, R. M. (1996). *Bayesian Learning for Neural Networks.* Springer. Lecture Notes in Statistics 118.

Williams, C. K. I. (1997). Computing with infinite networks. This volume.